# A Collapsed Variational Bayesian Inference Algorithm for Latent Dirichlet Allocation

**Yee Whye Teh**
Gatsby Computational Neuroscience Unit
University College London
17 Queen Square, London WC1N 3AR, UK
*ywteh@gatsby.ucl.ac.uk*

**David Newman and Max Welling**
Bren School of Information and Computer Science
University of California, Irvine
CA 92697-3425 USA
{*newman,welling*}*@ics.uci.edu*

## Abstract

Latent Dirichlet allocation (LDA) is a Bayesian network that has recently gained much popularity in applications ranging from document modeling to computer vision. Due to the large scale nature of these applications, current inference procedures like variational Bayes and Gibbs sampling have been found lacking. In this paper we propose the collapsed variational Bayesian inference algorithm for LDA, and show that it is computationally efficient, easy to implement and significantly more accurate than standard variational Bayesian inference for LDA.

## 1 Introduction

Bayesian networks with discrete random variables form a very general and useful class of probabilistic models. In a Bayesian setting it is convenient to endow these models with Dirichlet priors over the parameters as they are conjugate to the multinomial distributions over the discrete random variables [1]. This choice has important computational advantages and allows for easy inference in such models.

A class of Bayesian networks that has gained significant momentum recently is latent Dirichlet allocation (LDA) [2], otherwise known as multinomial PCA [3]. It has found important applications in both text modeling [4, 5] and computer vision [6]. Training LDA on a large corpus of several million documents can be a challenge and crucially depends on an efficient and accurate inference procedure. A host of inference algorithms have been proposed, ranging from variational Bayesian (VB) inference [2], expectation propagation (EP) [7] to collapsed Gibbs sampling [5].

Perhaps surprisingly, the collapsed Gibbs sampler proposed in [5] seem to be the preferred choice in many of these large scale applications. In [8] it is observed that EP is not efficient enough to be practical while VB suffers from a large bias. However, collapsed Gibbs sampling also has its own problems: one needs to assess convergence of the Markov chain and to have some idea of mixing times to estimate the number of samples to collect, and to identify coherent topics across multiple samples. In practice one often ignores these issues and collects as many samples as is computationally feasible, while the question of topic identification is often sidestepped by using just 1 sample. Hence there still seems to be a need for more efficient, accurate and deterministic inference procedures.

In this paper we will leverage the important insight that a Gibbs sampler that operates in a collapsed space—where the parameters are marginalized out—mixes much better than a Gibbs sampler that samples parameters and latent topic variables simultaneously. This suggests that the parameters and latent variables are intimately coupled. As we shall see in the following, marginalizing out the parameters induces new dependencies between the latent variables (which are *conditionally* independent given the parameters), but these dependencies are spread out over many latent variables. This implies that the dependency between any two latent variables is expected to be small. This is

precisely the right setting for a mean field (i.e. fully factorized variational) approximation: a particular variable interacts with the remaining variables only through summary statistics called the field, and the impact of any single variable on the field is very small [9]. Note that this is not true in the joint space of parameters and latent variables because fluctuations in parameters can have a significant impact on latent variables. We thus conjecture that the mean field assumptions are much better satisfied in the collapsed space of latent variables than in the joint space of latent variables and parameters. In this paper we leverage this insight and propose a collapsed variational Bayesian (CVB) inference algorithm.

In theory, the CVB algorithm requires the calculation of very expensive averages. However, the averages only depend on sums of independent Bernoulli variables, and thus are very closely approximated with Gaussian distributions (even for relatively small sums). Making use of this approximation, the final algorithm is computationally efficient, easy to implement and significantly more accurate than standard VB.

## 2  Approximate Inference in Latent Dirichlet Allocation

LDA models each document as a mixture over topics. We assume there are $K$ latent topics, each being a multinomial distribution over a vocabulary of size $W$. For document $j$, we first draw a mixing proportion $\theta_j = \{\theta_{jk}\}$ over $K$ topics from a symmetric Dirichlet with parameter $\alpha$. For the $i$th word in the document, a topic $z_{ij}$ is drawn with topic $k$ chosen with probability $\theta_{jk}$, then word $x_{ij}$ is drawn from the $z_{ij}$th topic, with $x_{ij}$ taking on value $w$ with probability $\phi_{kw}$. Finally, a symmetric Dirichlet prior with parameter $\beta$ is placed on the topic parameters $\phi_k = \{\phi_{kw}\}$. The full joint distribution over all parameters and variables is:

$$p(\mathbf{x}, \mathbf{z}, \boldsymbol{\theta}, \boldsymbol{\phi}|\alpha, \beta) = \prod_{j=1}^{D} \frac{\Gamma(K\alpha)}{\Gamma(\alpha)^K} \prod_{k=1}^{K} \theta_{jk}^{\alpha-1+n_{jk\cdot}} \prod_{k=1}^{K} \frac{\Gamma(W\beta)}{\Gamma(\beta)^W} \prod_{w=1}^{W} \phi_{kw}^{\beta-1+n_{\cdot kw}} \tag{1}$$

where $n_{jkw} = \#\{i : x_{ij} = w, z_{ij} = k\}$, and dot means the corresponding index is summed out: $n_{\cdot kw} = \sum_j n_{jkw}$, and $n_{jk\cdot} = \sum_w n_{jkw}$.

Given the observed words $\mathbf{x} = \{x_{ij}\}$ the task of Bayesian inference is to compute the posterior distribution over the latent topic indices $\mathbf{z} = \{z_{ij}\}$, the mixing proportions $\boldsymbol{\theta} = \{\theta_j\}$ and the topic parameters $\boldsymbol{\phi} = \{\phi_k\}$. There are three current approaches, variational Bayes (VB) [2], expectation propagation [7] and collapsed Gibbs sampling [5]. We review the VB and collapsed Gibbs sampling methods here as they are the most popular methods and to motivate our new algorithm which combines advantages of both.

### 2.1  Variational Bayes

Standard VB inference upper bounds the negative log marginal likelihood $-\log p(\mathbf{x}|\alpha, \beta)$ using the variational free energy:

$$-\log p(\mathbf{x}|\alpha, \beta) \leq \widetilde{\mathcal{F}}(\tilde{q}(\mathbf{z}, \boldsymbol{\theta}, \boldsymbol{\phi})) = E_{\tilde{q}}[-\log p(\mathbf{x}, \mathbf{z}, \boldsymbol{\phi}, \boldsymbol{\theta}|\alpha, \beta)] - \mathcal{H}(\tilde{q}(\mathbf{z}, \boldsymbol{\theta}, \boldsymbol{\phi})) \tag{2}$$

with $\tilde{q}(\mathbf{z}, \boldsymbol{\theta}, \boldsymbol{\phi})$ an approximate posterior, $\mathcal{H}(\tilde{q}(\mathbf{z}, \boldsymbol{\theta}, \boldsymbol{\phi})) = E_{\tilde{q}}[-\log \tilde{q}(\mathbf{z}, \boldsymbol{\theta}, \boldsymbol{\phi})]$ the variational entropy, and $\tilde{q}(\mathbf{z}, \boldsymbol{\theta}, \boldsymbol{\phi})$ assumed to be fully factorized:

$$\tilde{q}(\mathbf{z}, \boldsymbol{\theta}, \boldsymbol{\phi}) = \prod_{ij} \tilde{q}(z_{ij}|\tilde{\gamma}_{ij}) \prod_{j} \tilde{q}(\theta_j|\tilde{\alpha}_j) \prod_{k} \tilde{q}(\phi_k|\tilde{\beta}_k) \tag{3}$$

$\tilde{q}(z_{ij}|\tilde{\gamma}_{ij})$ is multinomial with parameters $\tilde{\gamma}_{ij}$ and $\tilde{q}(\theta_j|\tilde{\alpha}_j)$, $\tilde{q}(\phi_k|\tilde{\beta}_k)$ are Dirichlet with parameters $\tilde{\alpha}_j$ and $\tilde{\beta}_k$ respectively. Optimizing $\widetilde{\mathcal{F}}(\tilde{q})$ with respect to the variational parameters gives us a set of updates guaranteed to improve $\widetilde{\mathcal{F}}(\tilde{q})$ at each iteration and converges to a local minimum:

$$\tilde{\alpha}_{jk} = \alpha + \sum_i \tilde{\gamma}_{ijk} \tag{4}$$

$$\tilde{\beta}_{kw} = \beta + \sum_{ij} \mathbf{1}(x_{ij}=w)\tilde{\gamma}_{ijk} \tag{5}$$

$$\tilde{\gamma}_{ijk} \propto \exp\left(\Psi(\tilde{\alpha}_{jk}) + \Psi(\tilde{\beta}_{kx_{ij}}) - \Psi(\textstyle\sum_w \tilde{\beta}_{kw})\right) \tag{6}$$

where $\Psi(y) = \frac{\partial \log \Gamma(y)}{\partial y}$ is the digamma function and $\mathbf{1}$ is the indicator function.

Although efficient and easily implemented, VB can potentially lead to very inaccurate results. Notice that the latent variables $\mathbf{z}$ and parameters $\boldsymbol{\theta}, \boldsymbol{\phi}$ can be strongly dependent in the true posterior $p(\mathbf{z}, \boldsymbol{\theta}, \boldsymbol{\phi}|\mathbf{x})$ through the cross terms in (1). This dependence is ignored in VB which assumes that latent variables and parameters are independent instead. As a result, the VB upper bound on the negative log marginal likelihood can be very loose, leading to inaccurate estimates of the posterior.

## 2.2 Collapsed Gibbs Sampling

Standard Gibbs sampling, which iteratively samples latent variables $\mathbf{z}$ and parameters $\boldsymbol{\theta}, \boldsymbol{\phi}$, can potentially have slow convergence due again to strong dependencies between the parameters and latent variables. Collapsed Gibbs sampling improves upon Gibbs sampling by marginalizing out $\boldsymbol{\theta}$ and $\boldsymbol{\phi}$ instead, therefore dealing with them exactly. The marginal distribution over $\mathbf{x}$ and $\mathbf{z}$ is

$$p(\mathbf{z}, \mathbf{x}|\alpha, \beta) = \prod_j \frac{\Gamma(K\alpha)}{\Gamma(K\alpha + n_{j..})} \prod_k \frac{\Gamma(\alpha + n_{jk.})}{\Gamma(\alpha)} \prod_k \frac{\Gamma(W\beta)}{\Gamma(W\beta + n_{.k.})} \prod_w \frac{\Gamma(\beta + n_{.kw})}{\Gamma(\beta)} \qquad (7)$$

Given the current state of all but one variable $z_{ij}$, the conditional probability of $z_{ij}$ is:

$$p(z_{ij} = k|\mathbf{z}^{\neg ij}, \mathbf{x}, \alpha, \beta) = \frac{(\alpha + n_{jk.}^{\neg ij})(\beta + n_{.kx_{ij}}^{\neg ij})(W\beta + n_{.k.}^{\neg ij})^{-1}}{\sum_{k'=1}^{K}(\alpha + n_{jk'.}^{\neg ij})(\beta + n_{.k'x_{ij}}^{\neg ij})(W\beta + n_{.k'.}^{\neg ij})^{-1}} \qquad (8)$$

where the superscript $\neg ij$ means the corresponding variables or counts with $x_{ij}$ and $z_{ij}$ excluded, and the denominator is just a normalization. The conditional distribution of $z_{ij}$ is multinomial with simple to calculate probabilities, so the programming and computational overhead is minimal.

Collapsed Gibbs sampling has been observed to converge quickly [5]. Notice from (8) that $z_{ij}$ depends on $z^{\neg ij}$ only through the counts $n_{jk.}^{\neg ij}, n_{.kx_{ij}}^{\neg ij}, n_{.k.}^{\neg ij}$. In particular, the dependence of $z_{ij}$ on any particular other variable $z_{i'j'}$ is very weak, especially for large datasets. As a result we expect the convergence of collapsed Gibbs sampling to be fast [10]. However, as with other MCMC samplers, and unlike variational inference, it is often hard to diagnose convergence, and a sufficiently large number of samples may be required to reduce sampling noise.

The argument of rapid convergence of collapsed Gibbs sampling is reminiscent of the argument for when mean field algorithms can be expected to be accurate [9]. The counts $n_{jk.}^{\neg ij}, n_{.kx_{ij}}^{\neg ij}, n_{.k.}^{\neg ij}$ act as fields through which $z_{ij}$ interacts with other variables. In particular, averaging both sides of (8) by $p(\mathbf{z}^{\neg ij}|\mathbf{x}, \alpha, \beta)$ gives us the Callen equations, a set of equations that the true posterior must satisfy:

$$p(z_{ij} = k|\mathbf{x}, \alpha, \beta) = E_{p(\mathbf{z}^{\neg ij}|\mathbf{x}, \alpha, \beta)}\left[\frac{(\alpha + n_{jk.}^{\neg ij})(\beta + n_{.kx_{ij}}^{\neg ij})(W\beta + n_{.k.}^{\neg ij})^{-1}}{\sum_{k'=1}^{K}(\alpha + n_{jk'.}^{\neg ij})(\beta + n_{.k'x_{ij}}^{\neg ij})(W\beta + n_{.k'.}^{\neg ij})^{-1}}\right] \qquad (9)$$

Since the latent variables are already weakly dependent on each other, it is possible to replace (9) by a set of mean field equations where latent variables are assumed independent and still expect these equations to be accurate. This is the idea behind the collapsed variational Bayesian inference algorithm of the next section.

## 3 Collapsed Variational Bayesian Inference for LDA

We derive a new inference algorithm for LDA combining the advantages of both standard VB and collapsed Gibbs sampling. It is a variational algorithm which, instead of assuming independence, models the dependence of the parameters on the latent variables in an exact fashion. On the other hand we still assume that latent variables are mutually independent. This is not an unreasonable assumption to make since as we saw they are only weakly dependent on each other. We call this algorithm collapsed variational Bayesian (CVB) inference.

There are two ways to deal with the parameters in an exact fashion, the first is to marginalize them out of the joint distribution and to start from (7), the second is to explicitly model the posterior of $\boldsymbol{\theta}, \boldsymbol{\phi}$ given $\mathbf{z}$ and $\mathbf{x}$ without any assumptions on its form. We will show that these two methods

are equivalent. The only assumption we make in CVB is that the latent variables $\mathbf{z}$ are mutually independent, thus we approximate the posterior as:

$$\hat{q}(\mathbf{z}, \boldsymbol{\theta}, \boldsymbol{\phi}) = \hat{q}(\boldsymbol{\theta}, \boldsymbol{\phi}|\mathbf{z}) \prod_{ij} \hat{q}(z_{ij}|\hat{\gamma}_{ij}) \tag{10}$$

where $\hat{q}(z_{ij}|\hat{\gamma}_{ij})$ is multinomial with parameters $\hat{\gamma}_{ij}$. The variational free energy becomes:

$$\widehat{\mathcal{F}}(\hat{q}(\mathbf{z})\hat{q}(\boldsymbol{\theta}, \boldsymbol{\phi}|\mathbf{z})) = E_{\hat{q}(\mathbf{z})\hat{q}(\boldsymbol{\theta}, \boldsymbol{\phi}|\mathbf{z})}[-\log p(\mathbf{x}, \mathbf{z}, \boldsymbol{\theta}, \boldsymbol{\phi}|\alpha, \beta)] - \mathcal{H}(\hat{q}(\mathbf{z})\hat{q}(\boldsymbol{\theta}, \boldsymbol{\phi}|\mathbf{z}))$$
$$= E_{\hat{q}(\mathbf{z})}[E_{\hat{q}(\boldsymbol{\theta}, \boldsymbol{\phi}|\mathbf{z})}[-\log p(\mathbf{x}, \mathbf{z}, \boldsymbol{\theta}, \boldsymbol{\phi}|\alpha, \beta)] - \mathcal{H}(\hat{q}(\boldsymbol{\theta}, \boldsymbol{\phi}|\mathbf{z}))] - \mathcal{H}(\hat{q}(\mathbf{z})) \tag{11}$$

We minimize the variational free energy with respect to $\hat{q}(\boldsymbol{\theta}, \boldsymbol{\phi}|\mathbf{z})$ first, followed by $\hat{q}(\mathbf{z})$. Since we do not restrict the form of $\hat{q}(\boldsymbol{\theta}, \boldsymbol{\phi}|\mathbf{z})$, the minimum is achieved at the true posterior $\hat{q}(\boldsymbol{\theta}, \boldsymbol{\phi}|\mathbf{z}) = p(\boldsymbol{\theta}, \boldsymbol{\phi}|\mathbf{x}, \mathbf{z}, \alpha, \beta)$, and the variational free energy simplifies to:

$$\widehat{\mathcal{F}}(\hat{q}(\mathbf{z})) \triangleq \min_{\hat{q}(\boldsymbol{\theta}, \boldsymbol{\phi}|\mathbf{z})} \widehat{\mathcal{F}}(\hat{q}(\mathbf{z})\hat{q}(\boldsymbol{\theta}, \boldsymbol{\phi}|\mathbf{z})) = E_{\hat{q}(\mathbf{z})}[-\log p(\mathbf{x}, \mathbf{z}|\alpha, \beta)] - \mathcal{H}(\hat{q}(\mathbf{z})) \tag{12}$$

We see that CVB is equivalent to marginalizing out $\boldsymbol{\theta}, \boldsymbol{\phi}$ before approximating the posterior over $\mathbf{z}$. As CVB makes a strictly weaker assumption on the variational posterior than standard VB, we have

$$\widehat{\mathcal{F}}(\hat{q}(\mathbf{z})) \leq \widetilde{\mathcal{F}}(\tilde{q}(\mathbf{z})) \triangleq \min_{\tilde{q}(\boldsymbol{\theta})\tilde{q}(\boldsymbol{\phi})} \widetilde{\mathcal{F}}(\tilde{q}(\mathbf{z})\tilde{q}(\boldsymbol{\theta})\tilde{q}(\boldsymbol{\phi})) \tag{13}$$

and thus CVB is a better approximation than standard VB. Finally, we derive the updates for the variational parameters $\hat{\gamma}_{ij}$. Minimizing (12) with respect to $\hat{\gamma}_{ijk}$, we get

$$\hat{\gamma}_{ijk} = \hat{q}(z_{ij} = k) = \frac{\exp\left(E_{\hat{q}(\mathbf{z}^{\neg ij})}[p(\mathbf{x}, \mathbf{z}^{\neg ij}, z_{ij} = k|\alpha, \beta)]\right)}{\sum_{k'=1}^{K} \exp\left(E_{\hat{q}(\mathbf{z}^{\neg ij})}[p(\mathbf{x}, \mathbf{z}^{\neg ij}, z_{ij} = k'|\alpha, \beta)]\right)} \tag{14}$$

Plugging in (7), expanding $\log \frac{\Gamma(\eta + n)}{\Gamma(\eta)} = \sum_{l=0}^{n-1} \log(\eta + l)$ for positive reals $\eta$ and positive integers $n$, and cancelling terms appearing both in the numerator and denominator, we get

$$\hat{\gamma}_{ijk} = \frac{\exp\left(E_{\hat{q}(\mathbf{z}^{\neg ij})}[\log(\alpha + n_{jk\cdot}^{\neg ij}) + \log(\beta + n_{\cdot kx_{ij}}^{\neg ij}) - \log(W\beta + n_{\cdot k\cdot}^{\neg ij})]\right)}{\sum_{k'=1}^{K} \exp\left(E_{\hat{q}(\mathbf{z}^{\neg ij})}[\log(\alpha + n_{jk'\cdot}^{\neg ij}) + \log(\beta + n_{\cdot k'x_{ij}}^{\neg ij}) - \log(W\beta + n_{\cdot k'\cdot}^{\neg ij})]\right)} \tag{15}$$

### 3.1 Gaussian approximation for CVB Inference

For completeness, we describe how to compute each expectation term in (15) exactly in the appendix. This exact implementation of CVB is computationally too expensive to be practical, and we propose instead to use a simple Gaussian approximation which works very accurately and which requires minimal computational costs.

In this section we describe the Gaussian approximation applied to $E_{\hat{q}}[\log(\alpha + n_{jk\cdot}^{\neg ij})]$; the other two expectation terms are similarly computed. Assume that $n_{j\cdot\cdot} \gg 0$. Notice that $n_{jk\cdot}^{\neg ij} = \sum_{i' \neq i} \mathbf{1}(z_{i'j} = k)$ is a sum of a large number independent Bernoulli variables $\mathbf{1}(z_{i'j} = k)$ each with mean parameter $\hat{\gamma}_{i'jk}$, thus it can be accurately approximated by a Gaussian. The mean and variance are given by the sum of the means and variances of the individual Bernoulli variables:

$$E_{\hat{q}}[n_{jk\cdot}^{\neg ij}] = \sum_{i' \neq i} \hat{\gamma}_{i'jk} \qquad \qquad \mathrm{Var}_{\hat{q}}[n_{jk\cdot}^{\neg ij}] = \sum_{i' \neq i} \hat{\gamma}_{i'jk}(1 - \hat{\gamma}_{i'jk}) \tag{16}$$

We further approximate the function $\log(\alpha + n_{jk\cdot}^{\neg ij})$ using a second-order Taylor expansion about $E_{\hat{q}}[n_{jk\cdot}^{\neg ij}]$, and evaluate its expectation under the Gaussian approximation:

$$E_{\hat{q}}[\log(\alpha + n_{jk\cdot}^{\neg ij})] \approx \log(\alpha + E_{\hat{q}}[n_{jk\cdot}^{\neg ij}]) - \frac{\mathrm{Var}_{\hat{q}}(n_{jk\cdot}^{\neg ij})}{2(\alpha + E_{\hat{q}}[n_{jk\cdot}^{\neg ij}])^2} \tag{17}$$

Because $E_{\hat{q}}[n_{jk\cdot}^{\neg ij}] \gg 0$, the third derivative is small and the Taylor series approximation is very accurate. In fact, we have found experimentally that the Gaussian approximation works very well

even when $n_{j..}$ is small. The reason is that we often have $\hat{\gamma}_{i'jk}$ being either close to 0 or 1 thus the variance of $n_{jk.}^{\neg ij}$ is small relative to its mean and the Gaussian approximation will be accurate. Finally, plugging (17) into (15), we have our CVB updates:

$$\hat{\gamma}_{ijk} \propto \left(\alpha + E_{\hat{q}}[n_{jk.}^{\neg ij}]\right)\left(\beta + E_{\hat{q}}[n_{.kx_{ij}}^{\neg ij}]\right)\left(W\beta + E_{\hat{q}}[n_{.k.}^{\neg ij}]\right)^{-1}$$

$$\exp\left(-\frac{\mathrm{Var}_{\hat{q}}(n_{jk.}^{\neg ij})}{2(\alpha+E_{\hat{q}}[n_{jk.}^{\neg ij}])^2} - \frac{\mathrm{Var}_{\hat{q}}(n_{.kx_{ij}}^{\neg ij})}{2(\beta+E_{\hat{q}}[n_{.kx_{ij}}^{\neg ij}])^2} + \frac{\mathrm{Var}_{\hat{q}}(n_{.k.}^{\neg ij})}{2(W\beta+E_{\hat{q}}[n_{.k.}^{\neg ij}])^2}\right) \quad (18)$$

Notice the striking correspondence between (18), (8) and (9), showing that CVB is indeed the mean field version of collapsed Gibbs sampling. In particular, the first line in (18) is obtained from (8) by replacing the fields $n_{jk.}^{\neg ij}$, $n_{.kx_{ij}}^{\neg ij}$ and $n_{.k.}^{\neg ij}$ by their means (thus the term mean field) while the exponentiated terms are correction factors accounting for the variance in the fields.

CVB with the Gaussian approximation is easily implemented and has minimal computational costs. By keeping track of the mean and variance of $n_{jk.}$, $n_{.kw}$ and $n_{.k.}$, and subtracting the mean and variance of the corresponding Bernoulli variables whenever we require the terms with $x_{ij}, z_{ij}$ removed, the computational cost scales only as $O(K)$ for each update to $\hat{q}(z_{ij})$. Further, we only need to maintain one copy of the variational posterior over the latent variable for each unique document/word pair, thus the overall computational cost per iteration of CVB scales as $O(MK)$ where $M$ is the total number of unique document/word pairs, while the memory requirement is $O(MK)$. This is the same as for VB. In comparison, collapsed Gibbs sampling needs to keep track of the current sample of $z_{ij}$ for every word in the corpus, thus the memory requirement is $O(N)$ while the computational cost scales as $O(NK)$ where $N$ is the total number of words in the corpus—higher than for VB and CVB. Note however that the constant factor involved in the $O(NK)$ time cost of collapsed Gibbs sampling is significantly smaller than those for VB and CVB.

## 4 Experiments

We compared the three algorithms described in the paper: standard VB, CVB and collapsed Gibbs sampling. We used two datasets: first is "KOS" (www.dailykos.com), which has $J = 3430$ documents, a vocabulary size of $W = 6909$, a total of $N = 467,714$ words in all the documents and on average 136 words per document. Second is "NIPS" (books.nips.cc) with $J = 1675$ documents, a vocabulary size of $W = 12419$, $N = 2,166,029$ words in the corpus and on average 1293 words per document. In both datasets stop words and infrequent words were removed. We split both datasets into a training set and a test set by assigning 10% of the words in each document to the test set. In all our experiments we used $\alpha = 0.1$, $\beta = 0.1$, $K = 8$ number of topics for KOS and $K = 40$ for NIPS. We ran each algorithm on each dataset 50 times with different random initializations.

Performance was measured in two ways. First using variational bounds of the log marginal probabilities on the training set, and secondly using log probabilities on the test set. Expressions for the variational bounds are given in (2) for VB and (12) for CVB. For both VB and CVB, test set log probabilities are computed as:

$$p(\mathbf{x}^{\text{test}}) = \prod_{ij}\sum_k \bar{\theta}_{jk}\bar{\phi}_{kx_{ij}^{\text{test}}} \qquad \bar{\theta}_{jk} = \frac{\alpha + E_q[n_{jk.}]}{K\alpha + E_q[n_{j..}]} \qquad \bar{\phi}_{kw} = \frac{\beta + E_q[n_{.kw}]}{W\beta + E_q[n_{.k.}]} \quad (19)$$

Note that we used estimated mean values of $\theta_{jk}$ and $\phi_{kw}$ [11]. For collapsed Gibbs sampling, given $S$ samples from the posterior, we used:

$$p(\mathbf{x}^{\text{test}}) = \prod_{ij}\sum_k \frac{1}{|S|}\sum_{s=1}^{S}\theta_{jk}^s\phi_{kx_{ij}^{\text{test}}}^s \qquad \theta_{jk}^s = \frac{\alpha + n_{jk.}^s}{K\alpha + n_{j..}^s} \qquad \phi_{kw}^s = \frac{\beta + n_{.kw}^s}{W\beta + n_{.k.}^s} \quad (20)$$

Figure 1 summarizes our results. We show both quantities as functions of iterations and as histograms of final values for all algorithms and datasets. CVB converged faster and to significantly better solutions than standard VB; this confirms our intuition that CVB provides much better approximations than VB. CVB also converged faster than collapsed Gibbs sampling, but Gibbs sampling attains a better solution in the end; this is reasonable since Gibbs sampling should be exact with

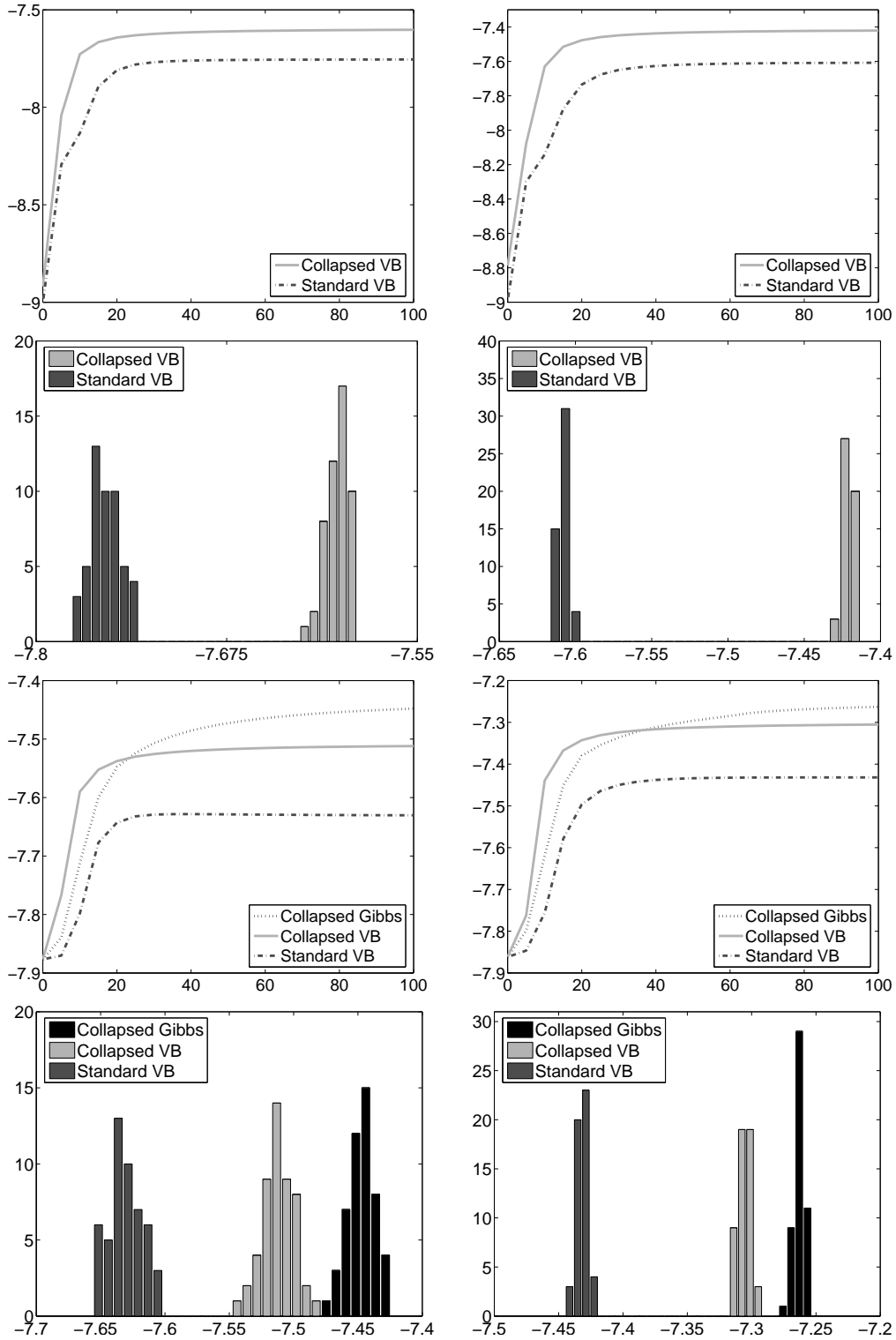

Figure 1: **Left**: results for KOS. **Right**: results for NIPS. **First row**: per word variational bounds as functions of numbers of iterations of VB and CVB. **Second row**: histograms of converged per word variational bounds across random initializations for VB and CVB. **Third row**: test set per word log probabilities as functions of numbers of iterations for VB, CVB and Gibbs. **Fourth row**: histograms of final test set per word log probabilities across 50 random initializations.

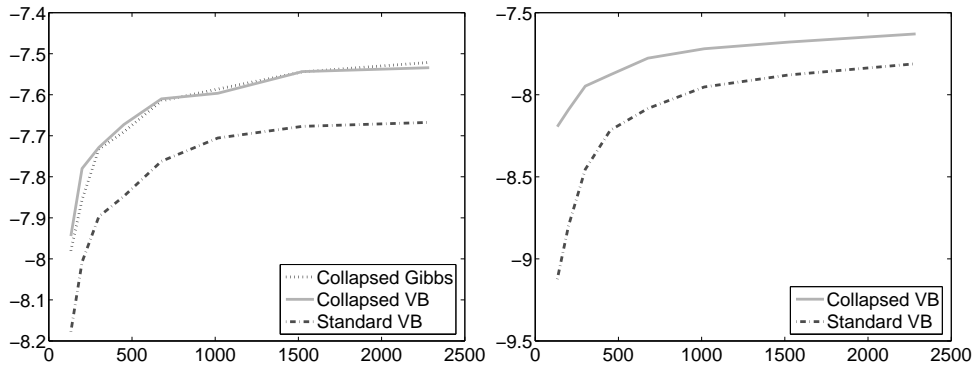

Figure 2: **Left**: test set per word log probabilities. **Right**: per word variational bounds. Both as functions of the number of documents for KOS.

enough samples. We have also applied the exact but much slower version of CVB without the Gaussian approximation, and found that it gave identical results to the one proposed here (not shown).

We have also studied the dependence of approximation accuracies on the number of documents in the corpus. To conduct this experiment we train on 90% of the words in a (growing) subset of the corpus and test on the corresponding 10% left out words. In figure Figure 2 we show both variational bounds and test set log probabilities as functions of the number of documents $J$. We observe that as expected the variational methods improve as $J$ increases. However, perhaps surprisingly, CVB does not suffer as much as VB for small values of $J$, even though one might expect that the Gaussian approximation becomes dubious in that regime.

# 5  Discussion

We have described a collapsed variational Bayesian (CVB) inference algorithm for LDA. The algorithm is easy to implement, computationally efficient and more accurate than standard VB. The central insight of CVB is that instead of assuming parameters to be independent from latent variables, we treat their dependence on the topic variables in an exact fashion. Because the factorization assumptions made by CVB are weaker than those made by VB, the resulting approximation is more accurate. Computational efficiency is achieved in CVB with a Gaussian approximation, which was found to be so accurate that there is never a need for exact summation.

The idea of integrating out parameters before applying variational inference has been independently proposed by [12]. Unfortunately, because they worked in the context of general conjugate-exponential families, the approach cannot be made generally computationally useful. Nevertheless, we believe the insights of CVB can be applied to a wider class of discrete graphical models beyond LDA. Specific examples include various extensions of LDA [4, 13] hidden Markov models with discrete outputs, and mixed-membership models with Dirichlet distributed mixture coefficients [14]. These models all have the property that they consist of discrete random variables with Dirichlet priors on the parameters, which is the property allowing us to use the Gaussian approximation. We are also exploring CVB on an even more general class of models, including mixtures of Gaussians, Dirichlet processes, and hierarchical Dirichlet processes.

Over the years a variety of inference algorithms have been proposed based on a combination of {maximize, sample, assume independent, marginalize out} applied to both parameters and latent variables. We conclude by summarizing these algorithms in Table 1, and note that CVB is located in the marginalize out parameters and assume latent variables are independent cell.

# A  Exact Computation of Expectation Terms in (15)

We can compute the expectation terms in (15) exactly as follows. Consider $E_{\hat{q}}[\log(\alpha + n_{jk\cdot}^{\neg ij})]$, which requires computing $\hat{q}(n_{jk\cdot}^{\neg ij})$ (other expectation terms are similarly computed). Note that

| Parameters → / ↓ Latent variables | maximize | sample | assume independent | marginalize out |
|---|---|---|---|---|
| maximize | Viterbi EM | ? | ME | ME |
| sample | stochastic EM | Gibbs sampling | ? | collapsed Gibbs |
| assume independent | variational EM | ? | VB | CVB |
| marginalize out | EM | any MCMC | EP for LDA | intractable |

Table 1: A variety of inference algorithms for graphical models. Note that not every cell is filled in (marked by ?) while some are simply intractable. "ME" is the maximization-expectation algorithm of [15] and "any MCMC" means that we can use any MCMC sampler for the parameters once latent variables have been marginalized out.

$n_{jk\cdot}^{\neg ij} = \sum_{i' \neq i} \mathbf{1}(z_{i'j} = k)$ is a sum of independent Bernoulli variables $\mathbf{1}(z_{i'j} = k)$ each with mean parameter $\hat{\gamma}_{i'jk}$. Define vectors $v_{i'jk} = [(1 - \hat{\gamma}_{i'jk}), \hat{\gamma}_{i'jk}]^{\top}$, and let $v_{jk} = v_{1jk} \otimes \cdots \otimes v_{n_{\cdot j}\cdot jk}$ be the convolution of all $v_{i'jk}$. Finally let $v_{jk}^{\neg ij}$ be $v_{jk}$ deconvolved by $v_{ijk}$. Then $\hat{q}(n_{jk\cdot}^{\neg ij} = m)$ will be the $(m+1)$st entry in $v_{jk}^{\neg ij}$. The expectation $E_{\hat{q}}[\log(\alpha + n_{jk\cdot}^{\neg ij})]$ can now be computed explicitly. This exact implementation requires an impractical $O(n_{j\cdot\cdot}^2)$ time to compute $E_{\hat{q}}[\log(\alpha + n_{jk\cdot}^{\neg ij})]$. At the expense of complicating the algorithm implementation, this can be improved by sparsifying the vectors $v_{jk}$ (setting small entries to zero) as well as other computational tricks. We propose instead the Gaussian approximation of Section 3.1, which we have found to give extremely accurate results but with minimal implementation complexity and computational cost.

## Acknowledgement

YWT was previously at NUS SoC and supported by the Lee Kuan Yew Endowment Fund. MW was supported by ONR under grant no. N00014-06-1-0734 and by NSF under grant no. 0535278.

## References

[1] D. Heckerman. A tutorial on learning with Bayesian networks. In M. I. Jordan, editor, *Learning in Graphical Models*. Kluwer Academic Publishers, 1999.

[2] D. M. Blei, A. Y. Ng, and M. I. Jordan. Latent Dirichlet allocation. *JMLR*, 3, 2003.

[3] W. Buntine. Variational extensions to EM and multinomial PCA. In *ECML*, 2002.

[4] M. Rosen-Zvi, T. Griffiths, M. Steyvers, and P. Smyth. The author-topic model for authors and documents. In *UAI*, 2004.

[5] T. L. Griffiths and M. Steyvers. Finding scientific topics. In *PNAS*, 2004.

[6] L. Fei-Fei and P. Perona. A Bayesian hierarchical model for learning natural scene categories. In *CVPR*, 2005.

[7] T. P. Minka and J. Lafferty. Expectation propagation for the generative aspect model. In *UAI*, 2002.

[8] W. Buntine and A. Jakulin. Applying discrete PCA in data analysis. In *UAI*, 2004.

[9] M. Opper and O. Winther. From naive mean field theory to the TAP equations. In D. Saad and M. Opper, editors, *Advanced Mean Field Methods : Theory and Practice*. The MIT Press, 2001.

[10] G. Casella and C. P. Robert. Rao-Blackwellisation of sampling schemes. *Biometrika*, 83(1):81–94, 1996.

[11] M. J. Beal. *Variational Algorithms for Approximate Bayesian Inference*. PhD thesis, Gatsby Computational Neuroscience Unit, University College London, 2003.

[12] J. Sung, Z. Ghahramani, and S. Choi. Variational Bayesian EM: A second-order approach. Unpublished manuscript, 2005.

[13] W. Li and A. McCallum. Pachinko allocation: DAG-structured mixture models of topic correlations. In *ICML*, 2006.

[14] E. M. Airoldi, D. M. Blei, E. P. Xing, and S. E. Fienberg. Mixed membership stochastic block models for relational data with application to protein-protein interactions. In *Proceedings of the International Biometrics Society Annual Meeting*, 2006.

[15] M. Welling and K. Kurihara. Bayesian K-means as a "maximization-expectation" algorithm. In *SIAM Conference on Data Mining*, 2006.
